# New Criteria and a New Algorithm for Learning in Multi-Agent Systems

**Rob Powers**
Computer Science Department
Stanford University
Stanford, CA 94305
powers@cs.stanford.edu

**Yoav Shoham**
Computer Science Department
Stanford University
Stanford, CA 94305
shoham@cs.stanford.edu

## Abstract

We propose a new set of criteria for learning algorithms in multi-agent systems, one that is more stringent and (we argue) better justified than previous proposed criteria. Our criteria, which apply most straightforwardly in repeated games with average rewards, consist of three requirements: (a) against a specified class of opponents (this class is a parameter of the criterion) the algorithm yield a payoff that approaches the payoff of the best response, (b) against other opponents the algorithm's payoff at least approach (and possibly exceed) the security level payoff (or maximin value), and (c) subject to these requirements, the algorithm achieve a close to optimal payoff in self-play. We furthermore require that these average payoffs be achieved quickly. We then present a novel algorithm, and show that it meets these new criteria for a particular parameter class, the class of stationary opponents. Finally, we show that the algorithm is effective not only in theory, but also empirically. Using a recently introduced comprehensive game theoretic test suite, we show that the algorithm almost universally outperforms previous learning algorithms.

## 1 Introduction

There is rapidly growing interest in multi-agent systems, and in particular in learning algorithms for such systems. There is a growing body of algorithms proposed, and some arguments about their relative merits and domains of applicability (for example, [14] and [17]). In [15] we survey much of this literature, and argue that it suffers from not having a clear objective criteria with which to evaluate each algorithm (this shortcoming is not unique to the relatively small computer science literature on multi-agent learning, and is shared by the much vaster literature on learning in game theory). In [15] we also define five different coherent agendas one could adopt, and identify one of them – the agent-centric one – as particularly relevant from the computer science point of view.

In the agent-centric agenda one asks how an agent can learn optimally in the presence of other independent agents, who may also be learning. To make the discussion precise we will concentrate on algorithms for learning in known, fully observable two-player repeated games, with average rewards. We start with the standard definition of a finite stage game (aka normal form game):

**Definition 1** *A two-player stage game is a tuple $G = (A_1, A_2, u_1, u_2)$, where*
- *$A_i$ is a finite set of actions available to player $i$*
- *$u_i : A_1 \times A_2 \to \Re$ is a utility function for player $i$*

Figure 1 shows two well-known games from the literature, to which we'll refer again later.

In a repeated game the stage game is repeated, finitely or infinitely. The agent accumulates rewards at each round; in the finite case the agent's aggregate reward is the average of the stage-game rewards, and in the infinite case it is the limit average (we ignore the subtlety that arises when the limit does not exist, but this case does not present an essential problem).

While the vast majority of the literature on multi-agent learning (surprisingly) does not start with a precise statement of objectives, there are some exceptions, and we discuss them in the next section, including their shortcomings. In the following section we propose a stronger set of criteria that, we believe, does not suffer from these limitations. We then present an algorithm that provably meets these stronger requirements. However, we believe that all formal requirements – including our own – are merely baseline guarantees, and any proposed algorithm must be subjected to empirical tests. While many previous proposals provide empirical results, we think it is fair to say that our level of empirical validation is unprecedented in the literature. We show the results of tests for all pairwise comparisons of major existing algorithms, using a recently-developed game theoretic testbed called GAMUT [13] to systematically sample a very large space of games.

## 2   Previous criteria for multi-agent learning

To our knowledge, Bowling and Veloso [1] were the first in the AI community to explicitly put forth formal requirements. Specifically they proposed two criteria:

**Rationality:** *If the other players' policies converge to stationary policies then the learning algorithm will converge to a stationary policy that is a best-response (in the stage game) to the other players' policies.*

**Convergence:** *The learner will necessarily converge to a stationary policy.*

Throughout this paper, we define a stationary policy as one that selects an action at each point during the game by drawing from the same distribution, regardless of past history.

Bowling and Veloso considered known repeated games and proposed an algorithm that provably meets their criteria in 2x2 games (games with two players and two actions per player). Later, Conitzer and Sandholm [5] adopted the same criteria, and demonstrated an algorithm meeting the criteria for all repeated games.

At first glance these criteria are reasonable, but a deeper look is less satisfying. First, note that the property of convergence cannot be applied unconditionally, since one cannot ensure that a learning procedure converges against all possible opponents without sacrificing rationality. So implicit in that requirement is some limitation on the class of opponents. And indeed both [1] and [5] acknowledge this and choose to concentrate on the case of self-play, that is, on opponents that are identical to the agent in question.

|         | $Dare$ | $Yield$ |
|---------|--------|---------|
| $Dare$  | $0, 0$ | $4, 1$  |
| $Yield$ | $1, 4$ | $2, 2$  |

|             | $Cooperate$ | $Defect$ |
|-------------|-------------|----------|
| $Cooperate$ | $3, 3$      | $0, 4$   |
| $Defect$    | $4, 0$      | $1, 1$   |

(a) Chicken                    (b) Prisoner's Dilemma

Figure 1: Example stage games. The payoff for the row player is given first in each cell, with the payoff for the column player following.

We will have more to say about self-play later, but there are other aspects of these criteria that bear discussion. While it is fine to consider opponents playing stationary policies, there are other classes of opponents that might be as relevant or even more relevant; this should be a degree of freedom in the definition of the problem. For instance, one might be interested in the classes of opponents that can be modeled by finite automata with at most $k$ states; these include both stationary and non-stationary strategies.

We find the property of requiring convergence to a stationary strategy particularly hard to justify. Consider the Prisoner's Dilemma game in Figure 1. The Tit-for-Tat algorithm[1] achieves an average payoff of 3 in self-play, while the unique Nash equilibrium of the stage game has a payoff of only 1. Similarly, in the game of Chicken, also shown in Figure 1, a strategy that alternates daring while its opponent yields and yielding while its opponent dares achieves a higher expected payoff in self-play than any stationary policy could guarantee. This problem is directly addressed in [2] and a counter-proposal made for how to consider equilibria in repeated games. But there is also a fundamental issue with these two criteria; they can both be thought of as a requirement on the *play* of the agent, rather than the *reward* the agent receives.

Our final point regarding these two criteria is that they express properties that hold in the limit, with no requirements whatsoever on the algorithm's performance in any finite period.

But this question is not new to the AI community and has been addressed numerous times in game theory, under the names of universal consistency, no-regret learning, and the Bayes envelope, dating back to [9] (see [6] for an overview of this history). There is a fundamental similarity in approach throughout, and we will take the two criteria proposed in [7] as being representative.

**Safety:** *The learning rule must guarantee at least the minimax payoff of the game. (The minimax payoff is the maximum expected value a player can guarantee against any possible opponent.)*

**Consistency:** *The learning rule must guarantee that it does at least as well as the best response to the empirical distribution of play when playing against an opponent whose play is governed by independent draws from any fixed distribution.*

They then define universal consistency as the requirement that a learning rule do at least as well as the best response to the empirical distribution regardless of the actual strategy the opponent is employing (this implies both safety and consistency) and show that a modification of the fictitious play algorithm [3] achieves this requirement. A limitation common to these game theory approaches is that they were designed for large-population games and therefore ignore the effect of the agent's play on the future play of the opponent. But this can pose problems in smaller games. Consider the game of Prisoner's Dilemma once again. Even if the opponent is playing Tit-for-Tat, the only universally consistent strategy would be to defect at every time step, ruling out the higher payoff achievable by cooperating.

## 3  A new set of criteria for learning

We will try to take the best of each proposal and create a joint set of criteria with the potential to address some of the limitations mentioned above.

We wish to keep the notion of optimality against a specific set of opponents. But instead of restricting this set in advance, we'll make this a parameter of the properties. Acknowledging that we may encounter opponents outside our target set, we will also incorporate the requirement of safety, which guarantees we achieve at least the security value, also known

as the maximin payoff, for the stage game. As a possible motivation for our approach, consider the game of Rock-Paper-Scissors, which despite its simplicity has motivated several international tournaments. While the unique Nash equilibrium policy is to randomize, the winners of the tournaments are those players who can most effectively exploit their opponents who deviate without being exploited in turn.

The question remains of how best to handle self-play. One method would be to require that a proposed algorithm be added to the set of opponents it is required to play a best response to. While this may seem appealing at first glance, it can be a very weak requirement on the actual payoff the agent receives. Since our opponent is no longer independent of our choice of strategy, we can do better than settling for just any mutual best response, and try to maximize the value we achieve as well. We therefore propose requiring the algorithm achieve at least the value of some Nash equilibrium that is Pareto efficient over the set of Nash equilibria.[2] Similarly, algorithms exist that satisfy 'universal consistency' and if played by all agents will converge to a correlated equilibria[10], but this result provides an even weaker constraint on the actual payoff received than convergence to a Nash equilibrium.

Let $k$ be the number of outcomes for the game and $b$ the maximum possible difference in payoffs across the outcomes. We require that for any choice of $\epsilon > 0$ and $\delta > 0$ there exist a $T_0$, polynomial in $\frac{1}{\epsilon}, \frac{1}{\delta}, k$, and $b$, such that for any number of rounds $t > T_0$ the algorithm achieves the following payoff guarantees with probability at least $1 - \delta$.

**Targeted Optimality:** *When the opponent is a member of the selected set of opponents, the average payoff is at least $V_{BR} - \epsilon$, where $V_{BR}$ is the expected value of the best response in terms of average payoff against the actual opponent.*

**Compatibility:** *During self-play, the average payoff is at least $V_{selfPlay} - \epsilon$, where $V_{selfPlay}$ is defined as the minimum value achieved by the player in any Nash equilibrium that is not Pareto dominated by another Nash equilibrium.*

**Safety:** *Against any opponent, the average payoff is at least $V_{security} - \epsilon$, with $V_{security}$ defined as $\max_{\pi_1 \in \Pi_1} \min_{\pi_2 \in \Pi_2} EV(\pi_1, \pi_2)$.[3]*

## 4  An algorithm

While we feel designing algorithms for use against more complex classes of opponent is critical, as a minimal requirement we first show an algorithm that meets the above criteria for the class of stationary opponents that has been the focus of much of the existing work. Our method incorporates modifications of three simple strategies: Fictitious Play [3], Bully [12], and the maximin strategy in order to create a more powerful hybrid algorithm.

Fictitious Play has been shown to converge in the limit to the best response against a stationary opponent. Each round it plays its best response to the most likely stationary opponent given the history of play. Our implementation uses a somewhat more generous best-response calculation so as to achieve our performance requirements during self-play.

$$BR_\epsilon(\pi) \leftarrow \underset{x \in X(\pi,\epsilon)}{\arg\max}(EOV(x, \pi)),^4$$

$$\text{where } X(\pi, \epsilon) = \{y \in \Pi_1 : EV(y, \pi) \geq \max_{z \in \Pi_1}(EV(z, \pi)) - \epsilon\}$$

We extend the Bully algorithm to consider the full set of mixed strategies and again maximize our opponent's value when multiple strategies yield equal payoff for our agent.

$$BullyMixed \leftarrow \arg\max_{x \in X}(EOV(x, BR(x))),$$

$$\text{where } X = \{y \in \Pi_1 : EV(y, BR_0(y)) = \max_{z \in \Pi_1}(EV(z, BR_0(z)))\}$$

The maximin strategy is defined as

$$Maximin \leftarrow \arg\max_{\pi_1 \in \Pi_1} \min_{\pi_2 \in \Pi_2} EV(\pi_1, \pi_2)$$

We will now show how to combine these strategies into a single method satisfying all three criteria. In the code shown below, $t$ is the current round, $AvgValue_m$ is the average value achieved by the agent during the last $m$ rounds, $V_{Bully}$ is shorthand for $EV(BullyMixed, BR_0(BullyMixed))$, and $d_{t_1}^{t_2}$ represents the distribution of opponent actions for the period from round $t_1$ to round $t_2$.

```
Set strategy = BullyMixed
for τ₁ time steps
  Play strategy
for τ₂ time steps
  if (strategy == BullyMixed AND AvgValueH < VBully − ε₁)
    With probability, p, set strategy = BRε₂(d₀ᵗ)
  Play strategy
if ||d₀^τ₁ − d_{t−τ₁}^t|| < ε₃
  Set bestStrategy = BRε₂(d₀ᵗ)
else if (strategy == BullyMixed AND AvgValueH > VBully − ε₁)
  Set bestStrategy = BullyMixed
else
  Set bestStrategy = BestResponse
while not end of game
  if avgValue_{t−τ₀} < Vsecurity − ε₀
    Play maximin strategy for τ₃ time steps
  else
    Play bestStrategy for τ₃ time steps
```

The algorithm starts out with a coordination/exploration period in which it attempts to determine what class its opponent is in. At the end of this period it chooses one of three strategies for the rest of the game. If it determines its opponent may be stationary it settles on a best response to the history up until that point. Otherwise, if the BullyMixed strategy has been performing well it maintains it. If neither of these conditions holds, it adopts a default strategy, which we have set to be the BestResponse strategy. This strategy changes each round, playing the best response to the maximum likelihood opponent strategy based on the last $H$ rounds of play. Once one of these strategies has been selected, the algorithm plays according to it whenever the average value meets or exceeds the security level, reverting to the maximin strategy if the value drops too low.

**Theorem 1** *Our algorithm satisfies the three properties stated in section 3 for the class of stationary opponents, with a $T_0$ proportional to $(\frac{b}{\epsilon})^3 \frac{1}{\delta}$.*

This theorem can be proven for all three properties using a combination of basic probability theory and repeated applications of the Hoeffding inequality [11], but the proof itself is prohibitively long for inclusion in this publication.

# 5 Empirical results

Although satisfying the criteria we put forth is comforting, we feel this is but a first step in making a compelling argument that an approach might be useful in practice. Traditionally, researchers suggesting a new algorithm also include an empirical comparison of the algorithm to previous work. While we think this is a critical component of evaluating an algorithm, most prior work has used tests against just one or two other algorithms on a very narrow set of test environments, which often vary from researcher to researcher. This practice has made it hard to consistently compare the performance of different algorithms.

In order to address this situation, we've started to code a collection of existing algorithms. Combining this set of algorithms with a wide variety of repeated games from GAMUT [13], a game theoretic test suite, we have the beginnings of a comprehensive testbed for multi-agent learning algorithms. In the rest of this section, we'll concentrate on the results for our algorithm, but we hope that this testbed can form the foundation for a broad, consistent framework of empirical testing in multi-agent learning going forward. For all of our environments we conducted our tests using a tournament format, where each algorithm plays all other algorithms including itself.

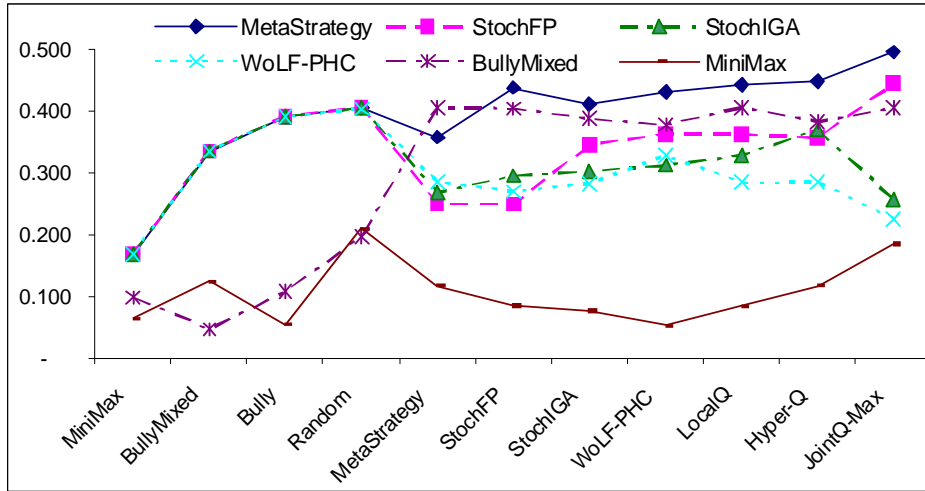

Figure 2: Average value for last 20K rounds (of 200K) across all games in GAMUT.

Let us first consider the results of a tournament over a full set of games in GAMUT. Figure 2 portrays the average value achieved by each agent (y-axis) averaged over all games, when playing different opponents (x-axis). The set of agents includes our strategy (MetaStrategy), six different adaptive learning approaches (Stochastic Fictitious Play [3,8], Stochastic IGA[16], WoLF-PHC[1], Hyper-Q learning[18], Local Q-learning[19], and JointQ-Max[4] (which learns Q-values over the joint action space but assumes its opponent will cooperate to maximize its payoff)), and four fixed strategies (BullyMixed, Bully[12], the maximin strategy, and Random (which selects a stationary mixed strategy at random)). We have chosen a subset of the most successful algorithms to display on the graph. Against the four stationary opponents, all of the adaptive learners fared equally well, while fixed strategy players achieved poor rewards. In contrast, BullyMixed fared well against the adaptive algorithms. As desired, our new algorithm combined the best of these characteristics to achieve the highest value against all opponents except itself. It fares worse than BullyMixed since it will always yield to BullyMixed, giving away the more advantageous outcome in games like Chicken. However, when comparing how each agent performs in self-play, our algorithm scores quite well, finishing a close second to Hyper-Q learning while the

two Bully algorithms finish near last. Hyper-Q is able to gain in self-play by occasionally converging to outcomes with high social welfare that our strategy does not consider.

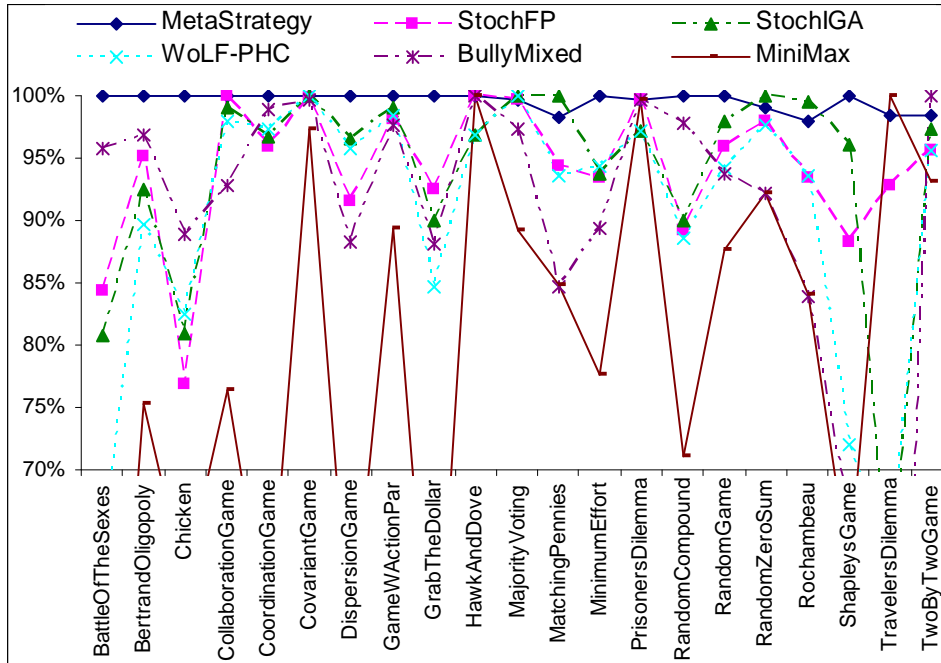

Figure 3: Percent of maximum value for last 20K rounds (of 200K) averaged across all opponents for selected games in GAMUT. The rewards were divided by the maximum reward achieved by any agent to make visual comparisons easier.

So far we've seen that our new algorithm performs well when playing against a variety of opponents. In Figure 3 we show the reward for each agent, averaged across the set of possible opponents for a selection of games in GAMUT. Once again our algorithm outperforms the existing algorithms in nearly all games. When it fails to achieve the highest reward it often appears to be due to its policy of "generosity"; in games where it has multiple actions yielding equal value, it chooses a best response that maximizes its opponent's value.

The ability to study how individual strategies fare in each class of environment reflects an advantage of our more comprehensive testing approach. In future work, this data can be used both to aid in the selection of an appropriate algorithm for a new environment and to pinpoint areas where an algorithm might be improved. Note that we use environment here to indicate a combination of both the game and the distribution over opponents.

## 6   Conclusions and Future Work

Our objective in this work was to put forth a new set of criteria for evaluating the performance of multi-agent learning algorithms as well as propose a more comprehensive method for empirical testing. In order to motivate this new approach for vetting algorithms, we have presented a novel algorithm that meets our criteria and outperforms existing algorithms in a wide variety of environments. We are continuing to work actively to extend our approach. In particular, we wish to demonstrate the generality of our approach by providing algorithms that calculate best response to different sets of opponents (conditional strategies, finite automata, etc.) Additionally, the criteria need to be generalized for $n$-player games

and we hope to combine our method for known games with methods for learning the structure of the game, ultimately devising new algorithms for unknown stochastic games.

## Acknowledgements

This work was supported in part by a Benchmark Stanford Graduate Fellowship, DARPA grant F30602-00-2-0598, and NSF grant IIS-0205633.

## Footnotes

[1]The Tit-for-Tat algorithm cooperates in the first round and then for each successive round plays the action its opponent played in the previous round.

[2] An outcome is Pareto efficient over a set if there is no other outcome in that set with a payoff at least as high for every agent and strictly higher for at least one agent.

[3] Throughout the paper, we use $EV(\pi_1, \pi_2)$ to indicate the expected payoff to a player for playing strategy $\pi_1$ against an opponent playing $\pi_2$ and $EOV(\pi_1, \pi_2)$ as the expected payoff the opponent achieves. $\Pi_1$ and $\Pi_2$ are the sets of mixed strategies for the agent and its opponent respectively.

[4] Note that $BR_0(\pi)$ is a member of the standard set of best response strategies to $\pi$.

## References

[1] Bowling, M. & Veloso, M. (2002). Multiagent learning using a variable learning rate. In *Artificial Intelligence, 136*, pp. 215-250.

[2] Brafman, R. & Tennenholtz, M. (2002). Efficient Learning Equilibrium. In *Advances in Neural Information Processing Systems* 15.

[3] Brown, G. (1951). Iterative Solution of Games by Fictitious Play. In *Activity Analysis of Production and Allocation*. New York: John Wiley and Sons.

[4] Claus, C. & Boutilier, C. (1998). The dynamics of reinforcement learning in cooperative multiagent systems. In *Proceedings of the National Conference on Artificial Intelligence* , pp. 746-752.

[5] Conitzer, V. & Sandholm, T. (2003). AWESOME: A General Multiagent Learning Algorithm that Converges in Self-Play and Learns a Best Response Against Stationary Opponents. In *Proceedings of the 20th International Conference on Machine Learning*, pp. 83-90, Washington, DC.

[6] Foster, D. & Vohra, R. (1999). Regret in the on-line decision problem. *"Games and Economic Behavior"* 29:7-36.

[7] Fudenberg, D. & Levine, D. (1995) Universal consistency and cautious fictitious play. *Journal of Economics Dynamics and Control* 19:1065-1089.

[8] Fudenberg, D. & Levine, D. (1998). *The theory of learning in games*. MIT Press.

[9] Hannan, J. (1957) Approximation to Bayes risk in repeated plays. *Contributions to the Theory of Games* 3:97-139.

[10] Hart, S. & Mas-Colell, A. (2000). A simple adaptive procedure leading to correlated equilibrium. In *Econometrica*, Vol. 68, No. 5, pages 1127-1150.

[11] Hoeffding, W. (1956). On the distribution of the number of successes in independent trials. *Annals of Mathematical Statistics* 27:713-721.

[12] Littman, M. & Stone, P. (2001). Implicit Negotiation in Repeated Games. In *Proceedings of the Eighth International Workshop on Agent Theories, Architectures, and Languages*, pp. 393-404.

[13] Nudelman, E., Wortman, J., Leyton-Brown, K., & Shoham, Y. (2004). Run the GAMUT: A Comprehensive Approach to Evaluating Game-Theoretic Algorithms. *AAMAS-2004*. To Appear.

[14] Sen, S. & Weiss, G. (1998). Learning in multiagent systems. In *Multiagent systems: A modern introduction to distributed artificial intelligence*, chapter 6, pp. 259-298, MIT Press.

[15] Shoham, Y., Powers, R., & Grenager, T. (2003). Multi-Agent Reinforcement Learning: a critical survey. Technical Report.

[16] Singh, S., Kearns, M., & Mansour, Y. (2000). Nash convergence of gradient dynamics in general-sum games. In *Proceedings of UAI-2000*, pp. 541-548, Morgan Kaufman.

[17] Stone, P. & Veloso, M. (2000). Multiagent systems: A survey from a machine learning perspective. *Autonomous Robots*, 8(3).

[18] Tesauro, G. (2004). Extending Q-Learning to General Adaptive Multi-Agent Systems. In *Advances in Neural Information Processing Systems 16*.

[19] Watkins, C. & Dayan, P. (1992). Technical note: Q-learning. *Machine Learning*, 8(3):279-292.
